# Incremental Parsing by Modular Recurrent Connectionist Networks

**Ajay N. Jain    Alex H. Waibel**
School of Computer Science
Carnegie Mellon University
Pittsburgh, PA 15213

## ABSTRACT

We present a novel, modular, recurrent connectionist network architecture which learns to robustly perform incremental parsing of complex sentences. From sequential input, one word at a time, our networks learn to do semantic role assignment, noun phrase attachment, and clause structure recognition for sentences with passive constructions and center embedded clauses. The networks make syntactic and semantic predictions at every point in time, and previous predictions are revised as expectations are affirmed or violated with the arrival of new information. Our networks induce their own "grammar rules" for dynamically transforming an input sequence of words into a syntactic/semantic interpretation. These networks generalize and display tolerance to input which has been corrupted in ways common in spoken language.

## 1    INTRODUCTION

Previously, we have reported on experiments using connectionist models for a small parsing task using a new network formalism which extends back-propagation to better fit the needs of sequential symbolic domains such as parsing (Jain, 1989). We showed that connectionist networks could learn the complex dynamic behavior needed in parsing. The task included passive sentences which require dynamic incorporation of previously unseen right context information into partially built syntactic/semantic interpretations. The trained parsing network exhibited predictive behavior and was able to modify or confirm

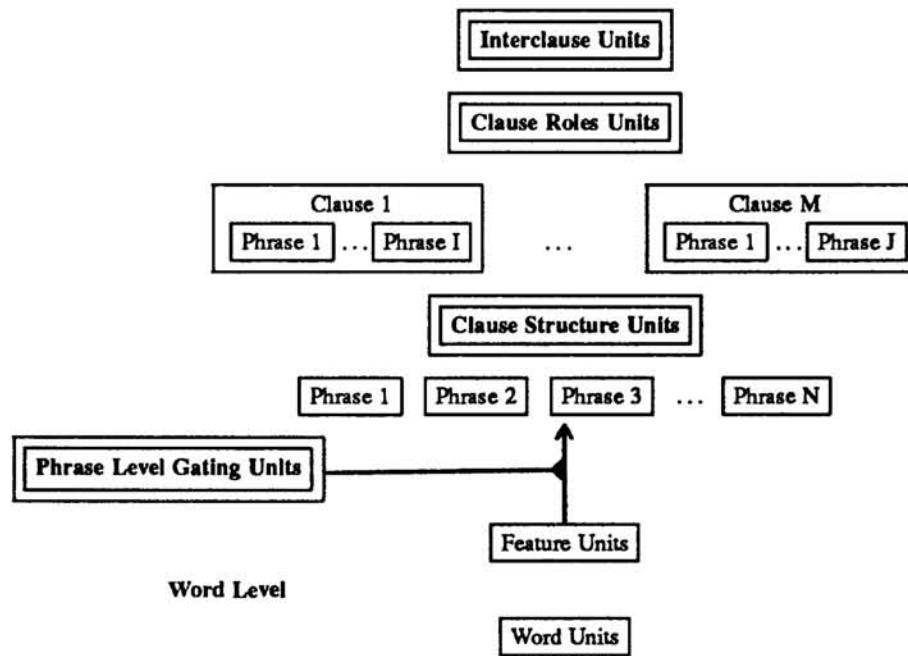

**Figure 1:** High-level Parsing Architecture.

hypotheses as sentences were sequentially processed. It was also able to generalize well and tolerate ill-formed input.

In this paper, we describe work on extending our parsing architecture to grammatically complex sentences.[1] The paper is organized as follows. First, we briefly outline the network formalism and the general architecture. Second, the parsing task is defined and the procedure for constructing and training the parser is presented. Then the dynamic behavior of the parser is illustrated, and the performance is characterized.

## 2  NETWORK ARCHITECTURE

We have developed an extension to back-propagation networks which is specifically designed to perform tasks in sequential domains requiring symbol manipulation (Jain, 1989). It is substantially different from other connectionist approaches to sequential problems (e.g. Elman, 1988; Jordan, 1986; Waibel *et al.*, 1989). There are four major features of this formalism. One, units retain partial activation between updates. They can respond to repetitive weak stimuli as well as singular sharp stimuli. Two, units are responsive to both static activation values of other units and their dynamic changes. Three, well-behaved symbol buffers can be constructed using groups of units whose connections are *gated* by other units. Four, the formalism supports recurrent networks. The networks are able to learn complex time-varying behavior using a gradient descent procedure via error back-propagation.

Figure 1 shows a high-level diagram of the general parsing architecture. It is organized into five hierarchical levels: Word, Phrase, Clause Structure, Clause Roles, and Inter-

clause. The description will proceed bottom up. A word is presented to the network by stimulating its associated word unit for a short time. This produces a pattern of activation across the feature units which represents the meaning of the word. The connections from the word units to the feature units which encode semantic and syntactic information about words are compiled into the network and are fixed.[2] The Phrase level uses the sequence of word representations from the Word level to build contiguous phrases. Connections from the Word level to the Phrase level are modulated by gating units which learn the required conditional assignment behavior. The Clause Structure level maps phrases into the constituent clauses of the input sentence. The Clause Roles level describes the roles and relationships of the phrases in each clause of the sentence. The final level, Interclause, represents the interrelationships among the clauses. The following section defines a parsing task and gives a detailed description of the construction and training of a parsing network which performs the task.

## 3   INCREMENTAL PARSING

In parsing spoken language, it is desirable to process input one word at a time as words are produced by the speaker and to incrementally build an output representation. This allows tight bi-directional coupling of the parser to the underlying speech recognition system. In such a system, the parser processes information as soon as it is produced and provides predictive information to the recognition system based on a rich representation of the current context. As mentioned earlier, our previous work applying connectionist architectures to a parsing task was promising. The experiment described below extends our previous work to grammatically complex sentences requiring a significant scale increase.

### 3.1   Parsing Task

The domain for the experiment was sentences with up to three clauses including nontrivial center-embedding and passive constructions.[3] Here are some example sentences:

- Fido dug up a bone near the tree in the garden.

- I know the man who John says Mary gave the book.

- The dog who ate the snake was given a bone.

Given sequential input, one word at a time, the task is to incrementally build a representation of the input sentence which includes the following information: phrase structure, clause structure, semantic role assignment, and interclause relationships. Figure 2 shows a representation of the desired parse of the last sentence in the list above.

```
[Clause 1:    [The dog RECIP] [was given ACTION] [a bone PATIENT]]
[Clause 2:    [who AGENT] [ate ACTION] [the snake PATIENT]
              (RELATIVE to Clause 1, Phrase 1)]
```
**Figure 2:** Representation of an Example Sentence.

### 3.2 Constructing the Parser

The architecture for the network follows that given in Figure 1. The following paragraphs describe the detailed network structure bottom up. The constraints on the numbers of objects and labels are fixed for a particular network, but the architecture itself is scalable. Wherever possible in the network construction, modularity and architectural constraints have been exploited to minimize training time and maximize generalization. A network was constructed from three separate recurrent subnetworks trained to perform a portion of the parsing task on the training sentences. The performance of the full network will be discussed in detail in the next section.

The Phrase level contains three types of units: *phrase block* units, *gating* units, and *hidden* units. There are 10 phrase blocks, each being able to capture up to 4 words forming a phrase. The phrase blocks contain sets of units (called slots) whose target activation patterns correspond to word feature patterns of words in phrases. Each slot has an associated gating unit which learns to conditionally assign an activation pattern from the feature units of the Word level to the slot. The gating units have input connections from the hidden units. The hidden units have input connections from the feature units, gating units, and phrase block units. The direct recurrence between the gating and hidden units allows the gating units to learn to inhibit and compete with one another. The indirect recurrence arising from the connections between the phrase blocks and the hidden units provides the context of the current input word. The target activation values for each gating unit are dynamically calculated during training; each gating unit must learn to become active at the proper time in order to perform the phrasal parsing. Each phrase block with its associated gating and hidden units has its weights slaved to the other phrase blocks in the Phrase level. Thus, if a particular phrase construction is only present in one position in the training set, all of the phrase blocks still learn to parse the construction.

The Clause Roles level also has shared weights among separate clause modules. This level is trained by simulating the sequential building and mapping of clauses to sets of units containing the phrase blocks for each clause (see Figure 1). There are two types of units in this level: *labeling* units and *hidden* units. The labeling units learn to label the phrases of the clauses with semantic roles and attach phrases to other (within-clause) phrases. For each clause, there is a set of units which assigns role labels (agent, patient, recipient, action) to phrases. There is also a set of units indicating phrasal modification. The hidden units are recurrently connected to the labeling units to provide context and competition as with the Phrase level; they also have input connections from the phrase blocks of a single clause. During training, the targets for the labeling units are set at the beginning of the input presentation and remain static. In order to minimize global error across the training set, the units must learn to become active or inactive as soon as

possible in the input. This forces the network to learn to be predictive.

The Clause Structure and Interclause levels are trained simultaneously as a single module. There are three types of units at this level: *mapping*, *labeling*, and *hidden* units. The mapping units assign phrase blocks to clauses. The labeling units indicate relative clause and a subordinate clause relationships. The mapping and labeling units are recurrently connected to the hidden units which also have input connections from the phrase blocks of the Phrase level. The behavior of the Phrase level is simulated during training of this module. This module utilizes no weight sharing techniques. As with the Clause Roles level, the targets for the labeling and mapping units are set at the beginning of input presentation, thus inducing the same type of predictive behavior.

## 4   PARSING PERFORMANCE

The separately trained submodules described above were assembled into a single network which performs the full parsing task. No additional training was needed to fine-tune the full parsing network despite significant differences between actual subnetwork performance and the simulated subnetwork performance used during training. The network successfully modeled the large diverse training set. This section discusses three aspects of the parsing network's performance: dynamic behavior of the integrated network, generalization, and tolerance to noisy input.

### 4.1   Dynamic Behavior

The dynamic behavior of the network will be illustrated on the example sentence from Figure 2: "The dog who ate the snake was given a bone." This sentence was not in the training set. Due to space limitations, actual plots of network behavior will only be presented for a small portion of the network.

Initially, all of the units in the network are at their resting values. The units of the phrase blocks all have low activation. The word unit corresponding to "the" is stimulated, causing its word feature representation to become active across the feature units of the Word level. The gating unit associated with the slot 1 of phrase block 1 becomes active, and the feature representation of "the" is assigned to the slot; the gate closes as the next word is presented. The remaining words of the sentence are processed similarly, resulting in the final Phrase level representation shown in Figure 2. While this is occurring, the higher levels of the network are processing the evolving Phrase level representation.

The behavior of some of the mapping units of the Clause Structure Level is shown in Figure 3. Early in the presentation of the first word, the Clause Structure level hypothesizes that the first 4 phrase blocks will belong to the first clause—reflecting the dominance of single clause sentences in the training set. After "the" is assigned to the first phrase block, this hypothesis is revised. The network then believes that there is an embedded clause of 3 (possibly 4) phrases following the first phrase. This predictive behavior emerged spontaneously from the training procedure (a large majority of sentences in the training set beginning with a determiner had embedded clauses after the first phrase). The next two words ("dog who") confirm the network's expectation. The word "ate" allows the network to firmly decide on an embedded clause of 3 phrases within

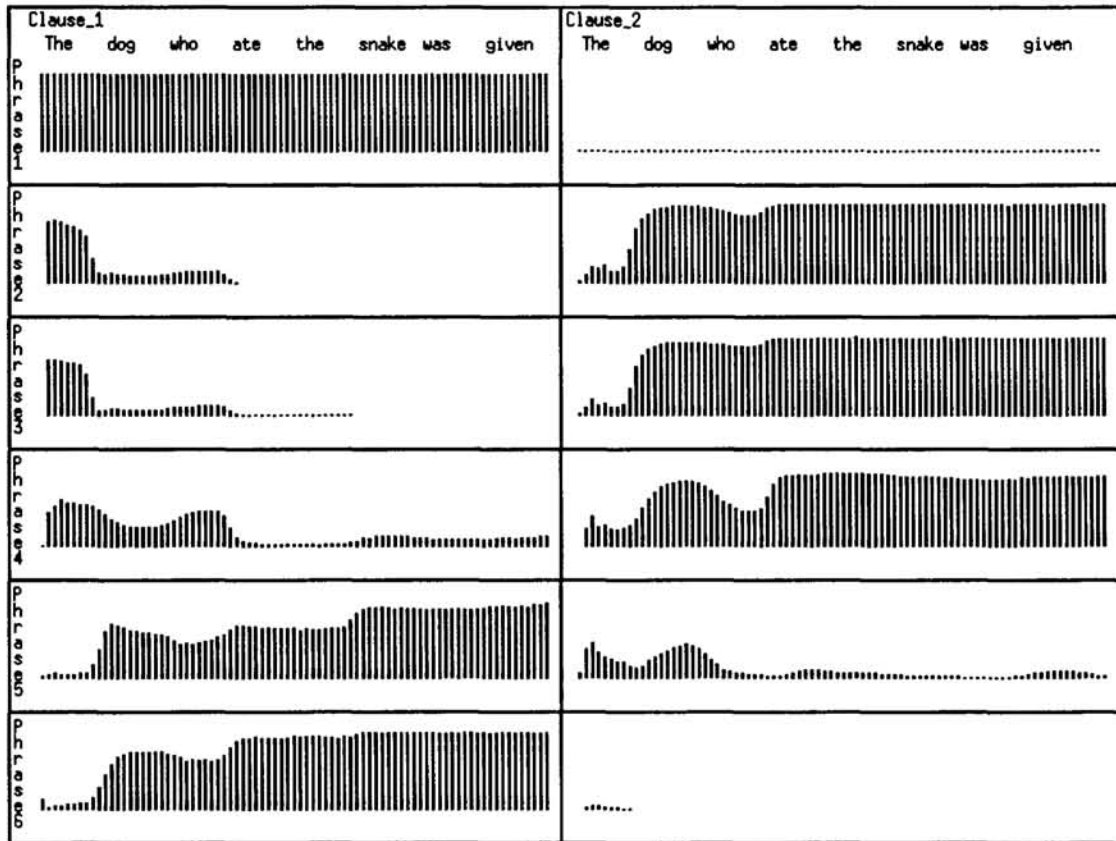

**Figure 3:** Example of Clause Structure Dynamic Behavior.

the main clause. This is the correct clausal structure of the sentence and is confirmed by the remainder of the input. The Interclause level indicates the appropriate relative clause relationship during the initial hypothesis of the embedded clause.

The Clause Roles level processes the individual clauses as they get mapped through the Clause Structure level. The labeling units for clause 1 initially hypothesize an Agent/Action/Patient role structure with some competition from a Rec/Act/Pat role structure (the Agent and Patient units' activation traces for clause 1, phrase 1 are shown in Figure 4). This prediction occurs because active constructs outnumbered passive ones during training. The final decision about role structure is postponed until just after the embedded clause is presented. The verb phrase "was given" immediately causes the Rec/Act/Pat role structure to dominate. Also, the network indicates that a fourth phrase (e.g. "by Mary") is expected to be the Agent. As with the first clause, an Ag/Act/Pat role structure is predicted for clause 2; this time the prediction is borne out.

## 4.2   Generalization

One type of generalization is automatic. A detail of the word representation scheme was omitted from the previous discussion. The feature patterns have two parts: a syntactic/semantic part and an identification part. The representations of "John" and "Peter" differ only in their ID parts. Units in the network which learn do not have any input connections from the ID portions of the word units. Thus, when the network learns to

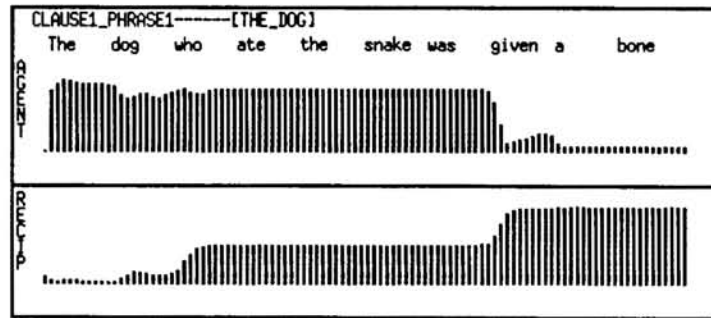

**Figure 4:** Example of Clause Roles Dynamic Behavior.

parse "John gave the bone to the dog," it will know how to parse "Peter promised the mitt to the boy." This type of generalization is extremely useful, both for addition of new words to the network and for processing many sentences not explicitly trained on.

The network also generalizes to correctly process truly novel sentences—sentences which are distinct (ignoring ID features) from those in the training set. The weight sharing techniques at the Phrase and Clause Structure levels have an impact here. While being difficult to measure generalization quantitatively, some statements can be made about the types of novel sentences which can be correctly processed relative to the training sentences. Substitution of single words resulting in a meaningful sentence is tolerated almost without exception. Substitution of entire phrases by different phrases causes some errors in structural parsing on sentences which have few similar training exemplars. However, the network does quite well on sentences which can be formed from composition between familiar sentences (e.g. interchanging clauses).

### 4.3    Tolerance to Noise

Several types of noise tolerance are interesting to analyze: ungrammaticality, word deletions (especially poorly articulated short function words), variance in word speed, inter-word silences, interjections, word/phrase repetitions, etc. The effects of noise were simulated by testing the parsing network on training sentences which had been corrupted in the ways listed above. Note that the parser was trained only on well-formed sentences.

Sentences in which verbs were made ungrammatical were processed without difficulty (e.g. "We am happy."). Sentences in which verb phrases were badly corrupted produced reasonable interpretations. For example, the sentence "Peter was gave a bone to Fido," received an Ag/Act/Pat/Rec role structure as if "was gave" was supposed to be either "gave" or "has given". Interpretation of corrupted verb phrases was context dependent.

Single clause sentences in which determiners were randomly deleted to simulate speech recognition errors were processed correctly 85 percent of the time. Multiple clause sentences degraded in a similar manner produced more parsing errors. There were fewer examples of multi-clause sentence types, and this hurt performance. Deletion of function words such as prepositions beginning prepositional phrases produced few errors, but deletions of critical function words such as "to" in infinitive constructions introducing subordinate clauses caused serious problems.

The network was somewhat sensitive to variations in word presentation speed (it was trained on a constant speed), but tolerated inter-word silences. Interjections of "ahh" and partial phrase repetitions were also tested. The network did not perform as well on these sentences as other networks trained for less complex parsing tasks. One possibility is that the weight sharing is preventing the formation of strong attractors for the training sentences. There appears to be a tradeoff between generalization and noise tolerance.

## 5   CONCLUSION

We have presented a novel connectionist network architecture and its application to a non-trivial parsing task. A hierarchical, modular, recurrent connectionist network was constructed which successfully learned to parse grammatically complex sentences. The parser exhibited predictive behavior and was able to dynamically revise hypotheses. Techniques for maximizing generalization were also discussed. Network performance on novel sentences was impressive. Results of testing the parser's sensitivity to several types of noise were somewhat mixed, but the parser performed well on ungrammatical sentences and sentences with non-critical function word deletions.

**Acknowledgments**

This research was funded by grants from ATR Interpreting Telephony Research Laboratories and the National Science Foundation under grant number EET-8716324. We thank Dave Touretzky for helpful comments and discussions.

**References**

J. L. Elman. (1988) *Finding Structure in Time.* Tech. Rep. 8801, Center for Research in Language, University of California, San Diego.

R. Hausser. (1988) *Computation of Language.* Springer-Verlag.

A. N. Jain. (1989) *A Connectionist Architecture for Sequential Symbolic Domains.* Tech. Rep. CMU-CS-89-187, School of Computer Science, Carnegie Mellon University.

A. N. Jain and A. H. Waibel. (1990) Robust connectionist parsing of spoken language. In *Proceedings of the 1990 IEEE International Conference on Acoustics, Speech, and Signal Processing.*

M. I. Jordan. (1986) *Serial Order: A Parallel Distributed Processing Approach.* Tech. Rep. 8604, Institute for Cognitive Science, University of California, San Diego.

R. Miikkulainen and M. G. Dyer. (1989) Encoding input/output representations in connectionist cognitive systems. In D. Touretzky, G. Hinton, and T. Sejnowski (eds.), *Proceedings of the 1988 Connectionist Models Summer School*, pp. 347–356, Morgan Kaufmann Publishers.

A. Waibel, T. Hanazawa, G. Hinton, K. Shikano, and K. Lang. (1989) Phoneme recognition using time-delay neural networks. *IEEE Transactions on Acoustics, Speech, and Signal Processing* 37(3):328–339.

## Footnotes

[1]Another presentation of this work appears in Jain and Waibel (1990).

[2]Connectionist networks have been used for lexical acquisition successfully (Miikkulainen and Dyer, 1989). However, in building large systems, it makes sense from an efficiency perspective to precompile as much lexical information as possible into a network. This is a pragmatic design choice in building large systems.

[3]The training set contained over 200 sentences. These are a subset of the sentences which form the example set of a parser based on a left associative grammar (Hausser, 1988). These sentences are grammatically interesting, but they do not reflect the statistical structure of common speech.
